# Source Separation with a Sensor Array Using Graphical Models and Subband Filtering

**Hagai Attias**
Microsoft Research
Redmond, WA 98052
hagaia@microsoft.com

## Abstract

Source separation is an important problem at the intersection of several fields, including machine learning, signal processing, and speech technology. Here we describe new separation algorithms which are based on probabilistic graphical models with latent variables. In contrast with existing methods, these algorithms exploit detailed models to describe source properties. They also use subband filtering ideas to model the reverberant environment, and employ an explicit model for background and sensor noise. We leverage variational techniques to keep the computational complexity per EM iteration linear in the number of frames.

## 1 The Source Separation Problem

Fig. 1 illustrates the problem of source separation with a sensor array. In this problem, signals from $K$ independent sources are received by each of $L \geq K$ sensors. The task is to extract the sources from the sensor signals. It is a difficult task, partly because the received signals are distorted versions of the originals. There are two types of distortions. The first type arises from propagation through a medium, and is approximately linear but also history dependent. This type is usually termed reverberations. The second type arises from background noise and sensor noise, which are assumed additive. Hence, the actual task is to obtain an *optimal estimate* of the sources from data. The task is difficult for another reason, which is lack of advance knowledge of the properties of the sources, the propagation medium, and the noises. This difficulty gave rise to adaptive source separation algorithms, where parameters that are related to those properties are adjusted to optimized a chosen cost function.

Unfortunately, the intense activity this problem has attracted over the last several years [1–9] has not yet produced a satisfactory solution. In our opinion, the reason is that existing techniques fail to address three major factors. The first is noise robustness: algorithms typically ignore background and sensor noise, sometime assuming they may be treated as additional sources. It seems plausible that to produce a noise robust algorithm, noise signals and their properties must be modeled explicitly, and these models should be exploited to compute optimal source estimators. The second factor is mixing filters: algorithms typically seek, and directly optimize, a transformation that would unmix the sources. However, in many situations, the filters describing medium propagation are non-invertible, or have an unstable inverse, or have a stable inverse that is extremely long. It may hence be advantageous to

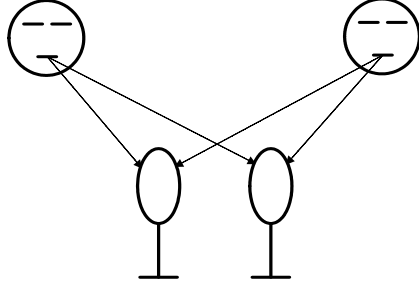

Figure 1: The source separation problem. Signals from $K = 2$ speakers propagate toward $L = 2$ sensors. Each sensor receives a linear mixture of the speaker signals, distorted by multipath propagation, medium response, and background and sensor noise. The task is to infer the original signals from sensor data.

estimate the mixing filters themselves, then use them to estimate the sources. The third factor is source properties: algorithms typically use a very simple source model (e.g., a one time point histogram). But in many cases one may easily obtain detailed models of the source signals. This is particularly true for speech sources, where large datasets exist and much modeling expertise has developed over decades of research. Separation of speakers is also one of the major potential commercial applications of source separation algorithms. It seems plausible that incorporating strong source models could improve performance. Such models may potentially have two more advantages: first, they could help limit the range of possible mixing filters by constraining the optimization problem. Second, they could help avoid whitening the extracted signals by effectively limiting their spectral range to the range characteristic of the source model.

This paper makes several contributions to the problem of real world source separation. In the following, we present new separation algorithms that are the first to address all three factors. We work in the framework of probabilistic graphical models. This framework allows us to construct models for sources and for noise, combine them with the reverberant mixing transformation in a principled manner, and compute parameter and source estimates from data which are Bayes optimal. We identify three technical ideas that are key to our approach: (1) a strong speech model, (2) subband filtering, and (3) variational EM.

## 2   Frames, Subband Signals, and Subband Filtering

We start with the concept of subband filtering. This is also a good point to define our notation. Let $x_m$ denote a time domain signal, e.g., the value of a sound pressure waveform at time point $m = 0, 1, 2, ....$. Let $X_n[k]$ denote the corresponding subband signal at time frame $n$ and subband frequency $k$. The subband signals are obtained from the time domain signal by imposing an $N$-point window $w_m, m = 0 : N - 1$ on that signal at equally spaced points $nJ$, $n = 0, 1, 2, ...$, and FFT-ing the windowed signal,

$$X_n[k] = \sum_{m=0}^{N-1} e^{-i\omega_k m} w_m x_{nJ+m} , \tag{1}$$

where $\omega_k = 2\pi k/N$ and $k = 0 : N - 1$. The subband signals are also termed *frames*. Notice the difference in time scale between the time frame index $n$ in $X_n[k]$ and the time point index $n$ in $x_n$.

The chosen value of the spacing $J$ depends on the window length $N$. For $J \leq N$ the original signal $x_m$ can be synthesized exactly from the subband signals (synthesis formula omitted).

An important consideration for selecting $J$, as well as the window shape, is behavior under filtering. Consider a filter $h_m$ applied to $x_m$, and denote by $y_m$ the filtered signal. In the simple case $h_m = h\delta_{m,0}$ (no filtering), the subband signals keep the same dependence as the time domain ones, $y_n = hx_n \longrightarrow Y_n[k] = hX_n[k]$. For an arbitrary filter $h_m$, we use the relation

$$y_n = \sum_m h_m x_{n-m} \longrightarrow Y_n[k] = \sum_m H_m[k]X_{n-m}[k] , \tag{2}$$

with complex coefficients $H_m[k]$ for each $k$. This relation between the subband signals is termed subband filtering, and the $H_m[k]$ are termed subband filters. Unlike the simple case of non-filtering, the relation (2) holds approximately, but quite accurately using an appropriate choice of $J$ and $w_m$; see [13] for details on accuracy. Throughout this paper, we will assume that an arbitrary filter $h_m$ can be modeled by the subband filters $H_m[k]$ to a sufficient accuracy for our purposes.

One advantage of subband filtering is that it replaces a long filter $h_m$ by a set of short independent filters $H_m[k]$, one per frequency. This will turn out to decompose the source separation problem into a set of small (albeit coupled) problems, one per frequency. Another advantage is that this representation allows using a detailed speech model on the same footing with the filter model. This is because a speech model is defined on the time scale of a single frame, whereas the original filter $h_m$, in contrast with $H_m[k]$, is typically as long as 10 or more frames.

As a final point on notation, we define a Gaussian distribution over a complex number $Z$ by $p(Z) = \mathcal{N}(Z \mid \mu, \nu) = \frac{\nu}{\pi}\exp(-\nu \mid Z - \mu \mid^2)$. Notice that this is a joint distribution over the real and imaginary parts of $Z$. The mean is $\mu = \langle X \rangle$ and the precision (inverse variance) $\nu$ satisfies $\nu^{-1} = \langle \mid X \mid^2 \rangle - \mid \mu \mid^2$.

# 3 A Model for Speech Signals

We assume independent sources, and model the distribution of source $j$ by a mixture model over its subband signals $X_{jn}$,

$$p(X_{jn} \mid S_{jn} = s) = \prod_{k=1}^{N/2-1} \mathcal{N}(X_{jn}[k] \mid 0, A_{js}[k]) \quad p(S_{jn} = s) = \pi_{js}$$

$$p(X, S) = \prod_{jn} p(X_{jn} \mid S_{jn})p(S_{jn}) , \tag{3}$$

where the components are labeled by $S_{jn}$. Component $s$ of source $j$ is a zero mean Gaussian with precision $A_{js}$. The mixing proportions of source $j$ are $\pi_{js}$. The DAG representing this model is shown in Fig. 2. A similar model was used in [10] for one microphone speech enhancement for recognition (see also [11]).

Here are several things to note about this model. (1) Each component has a characteristic spectrum, which may describe a particular part of a speech phoneme. This is because the precision corresponds to the inverse spectrum: the mean energy (w.r.t. the above distribution) of source $j$ at frequency $k$, conditioned on label $s$, is $\langle \mid X_{jn} \mid^2 \rangle = A_{js}^{-1}$. (2) A zero mean model is appropriate given the physics of the problem, since the mean of a sound pressure waveform is zero. (3) $k$ runs from 1 to $N/2 - 1$, since for $k > N/2$, $X_{jn}[k] = X_{jn}[N - k]^\star$; the subbands $k = 0, N/2$ are real and are omitted from the model, a common practice in speech recognition engines. (4) Perhaps most importantly, for each source *the subband signals are correlated* via the component label $s$, as $p(X_{jn}) = \sum_s p(X_{jn}, S_{jn} = s) \neq \prod_k p(X_{jn}[k])$. Hence, when the source separation problem decomposes into one problem per frequency, these problems turn out to be coupled (see below), and independent frequency permutations are avoided. (5) To increase

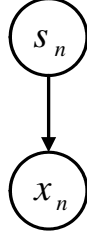

Figure 2: Graphical model describing speech signals in the subband domain. The model assumes i.i.d. frames; only the frame at time $n$ is shown. The node $X_n$ represents a complex $N/2 - 1$-dimensional vector $X_n[k]$, $k = 1 : N/2 - 1$.

model accuracy, a state transition matrix $p(S_{jn} = s \mid S_{j,n-1} = s')$ may be added for each source. The resulting HMM models are straightforward to incorporate without increasing the algorithm complexity.

There are several modes of using the speech model in the algorithms below. In one mode, the sources are trained online using the sensor data. In a second mode, source models are trained offline using available data on each source in the problem. A third mode correspond to separation of sources known to be speech but whose speakers are unknown. In this case, all sources have the same model, which is trained offline on a large dataset of speech signals, including 150 male and female speakers reading sentences from the Wall Street Journal (see [10] for details). This is the case presented in this paper. The training algorithm used was standard EM (omitted) using 256 clusters, initialized by vector quantization.

## 4    Separation of Non-Reverberant Mixtures

We now present a source separation algorithm for the case of non-reverberant (or instantaneous) mixing. Whereas many algorithms exist for this case, our contribution here is an algorithm that is significantly more robust to noise. Its robustness results, as indicated in the introduction, from three factors: (1) explicitly modeling the noise in the problem, (2) using a strong source model, in particular modeling the temporal statistics (over $N$ time points) of the sources, rather than one time point statistics, and (3) extracting each source signal from data by a Bayes optimal estimator obtained from $p(X \mid Y)$. A more minor point is handling the case of less sources than sensors in a principled way.

The mixing situation is described by $y_{in} = \sum_j h_{ij} x_{jn} + u_{in}$ , where $x_{jn}$ is source signal $j$ at time point $n$, $y_{in}$ is sensor signal $i$, $h_{ij}$ is the instantaneous mixing matrix, and $u_{in}$ is the noise corrupting sensor $i$'s signal. The corresponding subband signals satisfy $Y_{in}[k] = \sum_j h_{ij} X_{jn}[k] + U_{in}[k]$ .

To turn the last equation into a probabilistic graphical model, we assume that noise $i$ has precision (inverse spectrum) $B_i[k]$, and that noises at different sensors are independent (the latter assumption is often inaccurate but can be easily relaxed). This yields

$$
\begin{aligned}
p(Y_{in} \mid X) &= \prod_k \mathcal{N}(Y_{in}[k] \mid \sum_j h_{ij} X_{jn}[k], B_i[k]) \\
p(Y \mid X) &= \prod_{in} p(Y_{in} \mid X) ,
\end{aligned}
\tag{4}
$$

which together with the speech model (3) forms a complete model $p(Y, X, S)$ for this problem. The DAG representing this model for the case $K = L = 2$ is shown in Fig. 3. Notice that this model generalizes [4] to the subband domain.

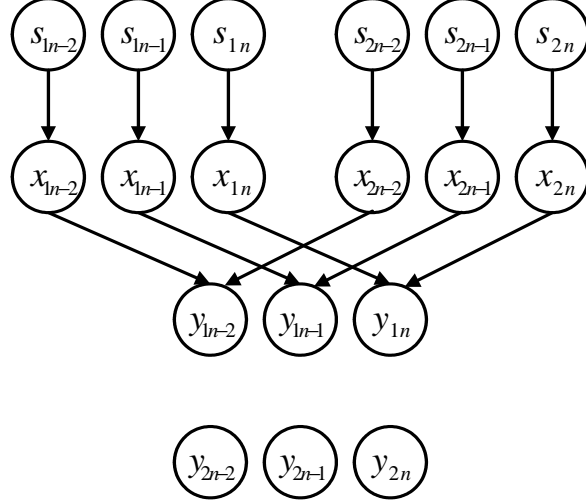

Figure 3: Graphical model for noisy, non-reverberant $2 \times 2$ mixing, showing a 3 frame-long sequence. All nodes $Y_{in}$ and $X_{jn}$ represent complex $N/2-1$-dimensional vectors (see Fig. 2). While $Y_{1n}$ and $Y_{2n}$ have the same parents, $X_{1n}$ and $X_{2n}$, the arcs from the parents to $Y_{2n}$ are omitted for clarity.

The model parameters $\theta = \{h_{ij}, B_i[k], A_{js}[k], \pi_{js}\}$ are estimated from data by an EM algorithm. However, as the number of speech components $M$ or the number of sources $K$ increases, the E-step becomes computationally intractable, as it requires summing over all $\mathcal{O}(M^K)$ configurations of $(S_{1n}, ..., S_{Kn})$ at each frame. We approximate the E-step using a variational technique: focusing on the posterior distribution $p(X, S \mid Y)$, we compute an optimal tractable approximation $q(X, S \mid Y) \approx p(X, S \mid Y)$, which we use to compute the sufficient statistics (SS). We choose

$$q(X, S \mid Y) = \prod_{jn} q(X_{jn} \mid S_{jn}, Y) q(S_{jn} \mid Y) , \qquad (5)$$

where the hidden variables are factorized over the sources, and also over the frames (the latter factorization is exact in this model, but is an approximation for reverberant mixing). This posterior maintains the dependence of $X$ on $S$, and thus the correlations between different subbands $X_{jn}[k]$. Notice also that this posterior implies a multimodal $q(X_{jn})$ (i.e., a mixture distribution), which is more accurate than unimodal posteriors often employed in variational approximations (e.g., [12]), but is also harder to compute. A slightly more general form which allows inter-frame correlations by employing $q(S \mid Y) = \prod_{jn} q(S_{jn} \mid S_{j,n-1}, Y)$ may also be used, without increasing complexity.

By optimizing in the usual way (see [12,13]) a lower bound on the likelihood w.r.t. $q$, we obtain

$$q(X_{jn}, S_{jn} = s \mid Y) = \prod_k q(X_{jn}[k] \mid S_{jn} = s, Y) q(S_{jn} = s \mid Y) , \qquad (6)$$

where $q(X_{jn}[k] \mid S_{jn} = s, Y) = \mathcal{N}(X_{jn}[k] \mid \rho_{jns}[k], \nu_{js}[k])$ and $q(S_{jn} = s \mid Y) = \gamma_{jns}$. Both the factorization over $k$ of $q(X_{jn} \mid S_{jn})$ and its Gaussian functional form fall out from the optimization under the structural restriction (5) and need not be specified in advance. The variational parameters $\{\rho_{jns}[k], \nu_{js}[k], \gamma_{jns}\}$, which depend on the data $Y$, constitute the SS and are computed in the E-step. The DAG representing this posterior is shown in Fig. 4.

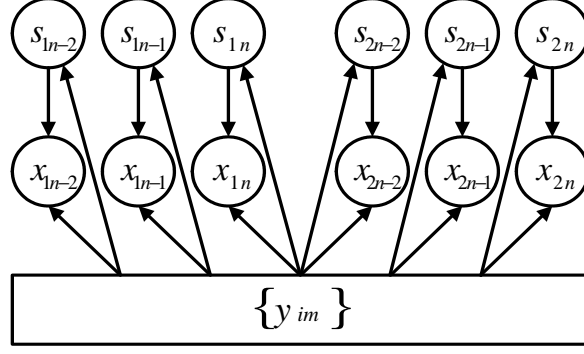

Figure 4: Graphical model describing the variational posterior distribution applied to the model of Fig. 3. In the non-reverberant case, the components of this posterior at time frame $n$ are conditioned only on the data $Y_{in}$ at that frame; in the reverberant case, the components at frame $n$ are conditioned on the data $Y_{im}$ at all frames $m$. For clarity and space reasons, this distinction is not made in the figure.

After learning, the sources are extracted from data by a variational approximation of the minimum mean squared error estimator,

$$\hat{X}_{jn}[k] = E(X_{jn}[k] \mid Y) = \int dX \; q(X \mid Y) X_{jn}[k] \; , \tag{7}$$

i.e., the posterior mean, where $q(X \mid Y) = \sum_S q(X, S \mid Y)$. The time domain waveform $\hat{x}_{jm}$ is then obtained by appropriately patching together the subband signals.

**M-step**. The update rule for the mixing matrix $h_{ij}$ is obtained by solving the linear equation

$$\sum_k B_i[k] \eta_{ij,0}[k] = \sum_{j'} h_{ij'} \sum_k B_i[k] \lambda_{j'j,0}[k] \; . \tag{8}$$

The update rule for the noise precisions $B_i[k]$ is omitted. The quantities $\eta_{ij,m}[k]$ and $\lambda_{j'j,m}[k]$ are computed from the SS; see [13] for details.

**E-step**. The posterior means of the sources (7) are obtained by solving

$$\hat{X}_{jn}[k] = \hat{\nu}_{jn}[k]^{-1} \sum_i B_i[k] h_{ij} \left( Y_{in}[k] - \sum_{j' \neq j} h_{ij'} \hat{X}_{j'n}[k] \right) \tag{9}$$

for $\hat{X}_{jn}[k]$, which is a $K \times K$ linear system for each frequency $k$ and frame $n$. The equations for the SS are given in [13], which also describes experimental results.

## 5   Separation of Reverberant Mixtures

In this section we extend the algorithm to the case of reverberant mixing. In that case, due to signal propagation in the medium, each sensor signal at time frame $n$ depends on the source signals not just at the same time but also at previous times. To describe this mathematically, the mixing matrix $h_{ij}$ must become a matrix of filters $h_{ij,m}$, and $y_{in} = \sum_{jm} h_{ij,m} x_{j,n-m} + u_{in}$.

It may seem straightforward to extend the algorithm derived above to the present case. However, this appearance is misleading, because we have a time scale problem. Whereas

are speech model $p(X, S)$ is frame based, the filters $h_{ij,m}$ are generally longer than the frame length $N$, typically 10 frames long and sometime longer. It is unclear how one can work with both $X_{jn}$ and $h_{ij,m}$ on the same footing (and, it is easy to see that straightforward windowed FFT cannot solve this problem).

This is where the idea of subband filtering becomes very useful. Using (2) we have $Y_{in}[k] = \sum_{jm} H_{ij,m}[k]X_{j,n-m}[k] + U_{in}[k]$, which yields the probabilistic model

$$p(Y_{in} \mid X) \quad = \quad \prod_k \mathcal{N}(Y_{in}[k] \mid \sum_{jm} H_{ij,m}[k]X_{j,n-m}[k], B_i[k]) \ . \qquad (10)$$

Hence, both $X$ and $Y$ are now frame based. Combining this equation with the speech model (3), we now have a complete model $p(Y, X, S)$ for the reverberant mixing problem. The DAG describing this model is shown in Fig. 5.

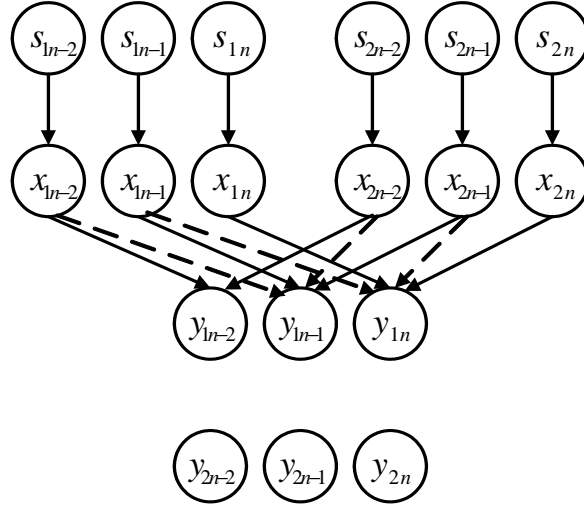

Figure 5: Graphical model for noisy, reverberant $2 \times 2$ mixing, showing a 3 frame-long sequence. Here we assume 2 frame-long filters, i.e., $m = 0, 1$ in Eq. (10), where the solid arcs from $X$ to $Y$ correspond to $m = 0$ (as in Fig. 3) and the dashed arcs to $m = 1$. While $Y_{1n}$ and $Y_{2n}$ have the same parents, $X_{1n}$ and $X_{2n}$, the arcs from the parents to $Y_{2n}$ are omitted for clarity.

The model parameters $\theta = \{H_{ij,m}[k], B_i[k], A_{js}[k], \pi_{js}\}$ are estimated from data by a variational EM algorithm, whose derivation generally follows the one outlined in the previous section. Notice that the exact E-step here is even more intractable, due to the history dependence introduced by the filters.

**M-step**. The update rule for $H_{ij,m}$ is obtained by solving the Toeplitz system

$$\sum_{j'm'} H_{ij',m'}[k]\lambda_{j'j,m-m'}[k] = \eta_{ij,m}[k] \qquad (11)$$

where the quantities $\lambda_{j'j,m}[k], \eta_{ij,m}[k]$ are computed from the SS (see [12]). The update rule for the $B_i[k]$ is omitted.

**E-step**. The posterior means of the sources (7) are obtained by solving

$$\hat{X}_{jn}[k] = \hat{\nu}_{jn}[k]^{-1} \sum_{im} B_i[k]H_{ij,m-n}[k]^\star \left( Y_{im}[k] - \sum_{j'm' \neq jm} H_{ij',m-m'}[k]\hat{X}_{j'm'}[k] \right) \quad (12)$$

for $\hat{X}_{jn}[k]$. Assuming $P$ frames long filters $H_{ij,m}$, $m = 0 : P - 1$, this is a $KP \times KP$ linear system for each frequency $k$. The equations for the SS are given in [13], which also describes experimental results.

# 6 Extensions

An alternative technique we have been pursuing for approximating EM in our models is Sequential Rao-Blackwellized Monte Carlo. There, we sample state sequences $S$ from the posterior $p(S \mid Y)$ and, for a given sequence, perform exact inference on the source signals $X$ conditioned on that sequence (observe that given $S$, the posterior $p(X \mid S, Y)$ is Gaussian and can be computed exactly). In addition, we are extending our speech model to include features such as pitch [7] in order to improve separation performance, especially in cases with less sensors than sources [7–9]. Yet another extension is applying model selection techniques to infer the number of sources from data in a dynamic manner.

### Acknowledgments

I thank Te-Won Lee for extremely valuable discussions.

### References

[1] A.J. Bell, T.J. Sejnowski (1995). An information maximisation approach to blind separation and blind deconvolution. *Neural Computation* 7, 1129-1159.

[2] B.A. Pearlmutter, L.C. Parra (1997). Maximum likelihood blind source separation: A context-sensitive generalization of ICA. *Proc. NIPS-96*.

[3] A. Cichocki, S.-I. Amari (2002). *Adaptive Blind Signal and Image Processing*. Wiley.

[4] H. Attias (1999). Independent Factor Analysis. *Neural Computation* 11, 803-851.

[5] T.-W. Lee et al. (2001) (Ed.). *Proc. ICA 2001*.

[6] S. Griebel, M. Brandstein (2001). Microphone array speech dereverberation using coarse channel modeling. *Proc. ICASSP 2001*.

[7] J. Hershey, M. Casey (2002). Audiovisual source separation via hidden Markov models. *Proc. NIPS 2001*.

[8] S. Roweis (2001). One Microphone Source Separation. *Proc. NIPS-00*, 793-799.

[9] G.-J. Jang, T.-W. Lee, Y.-H. Oh (2003). A probabilistic approach to single channel blind signal separation. *Proc. NIPS 2002*.

[10] H. Attias, L. Deng, A. Acero, J.C. Platt (2001). A new method for speech denoising using probabilistic models for clean speech and for noise. *Proc. Eurospeech 2001*.

[11] Ephraim, Y. (1992). Statistical model based speech enhancement systems. *Proc. IEEE* 80(10), 1526-1555.

[12] M.I. Jordan, Z. Ghahramani, T.S. Jaakkola, L.K. Saul (1999). An introduction to variational methods in graphical models. *Machine Learning* 37, 183-233.

[13] H. Attias (2003). New EM algorithms for source separation and deconvolution with a microphone array. *Proc. ICASSP 2003*.
